# Segmental Neural Net Optimization for Continuous Speech Recognition

Ying Zhao    Richard Schwartz    John Makhoul    George Zavaliagkos
BBN System and Technologies
70 Fawcett Street
Cambridge MA 02138

## Abstract

Previously, we had developed the concept of a Segmental Neural Net (SNN) for phonetic modeling in continuous speech recognition (CSR). This kind of neural network technology advanced the state-of-the-art of large-vocabulary CSR, which employs Hidden Markov Models (HMM), for the ARPA 1000-word Resource Management corpus. More Recently, we started porting the neural net system to a larger, more challenging corpus – the ARPA 20,000-word Wall Street Journal (WSJ) corpus. During the porting, we explored the following research directions to refine the system: i) training context-dependent models with a regularization method; ii) training SNN with projection pursuit; and ii) combining different models into a hybrid system. When tested on both a development set and an independent test set, the resulting neural net system alone yielded a performance at the level of the HMM system, and the hybrid SNN/HMM system achieved a consistent 10-15% word error reduction over the HMM system. This paper describes our hybrid system, with emphasis on the optimization methods employed.

## 1  INTRODUCTION

Hidden Markov Models (HMM) represent the state-of-the-art for large-vocabulary continuous speech recognition (CSR). Recently, neural network technology has been shown to advance the state-of-the-art for CSR by integrating neural nets and HMMs [1,2]. In principle, the advance is based on the fact that neural network modeling can avoid some limitations of the HMM modeling, for example, the conditional-independence assumption of HMMs and the fact that segmental features are hard to incorporate. Our work has been based on the concept of a Segmental Neural Net (SNN) [2].

A segmental neural network is a neural network that attempts to recognize a complete phoneme segment as a single unit. Its basic structure is shown in Figure 1. The input to the network is a fixed length representation of the speech segment, which is obtained from the warping (quasi-linear sampling) of a variable length phoneme segment. If the network is trained to minimize a least squares error or a cross entropy distortion measure, the output of the network can be shown to be an estimate of the posterior probability of the phoneme class given the input segment [3,4].

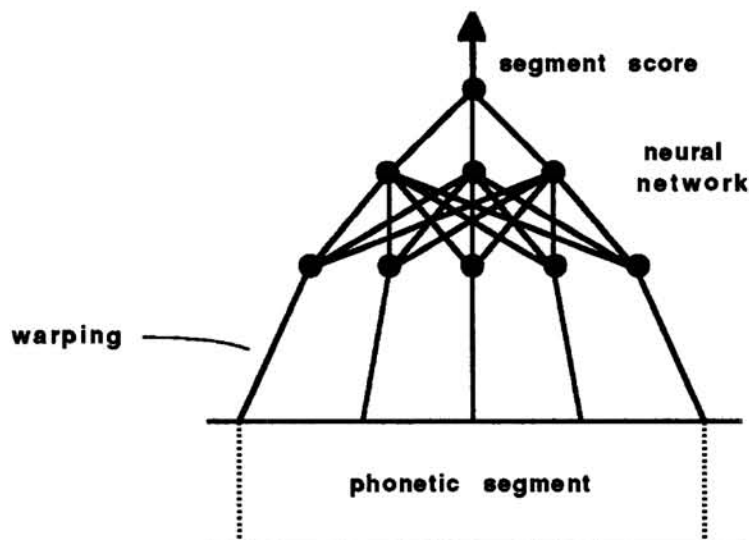

Figure 1: The SNN model samples the frames and produces a single segment score.

Our initial SNN system comprised a set of one-layer sigmoidal nets. This system is trained to minimize a cross entropy distortion measure by a quasi-Newton error minimization algorithm. A variable length segment is warped into a fixed length of 5 input frames. Since each frame includes 16 features, 14 mel cepstrum, power and difference of power, an input to the neural network forms a $16 \times 5 = 80$ dimensional vector.

Previously, our experimental domain was the ARPA 1000-word Resource Management (RM) Corpus, where we used 53 output phoneme classes. When tested on three independent evaluation sets (Oct 89, Feb 91 and Sep 92), our system achieved a consistent 10-20% word error rate reduction over the state-of-the-art HMM system [2].

## 2   THE WALL STREET JOURNAL CORPUS

After the final September 92 RM corpus evaluation, we ported our neural network system to a larger corpus — the Wall Street Journal (WSJ) Corpus. The WSJ corpus consists primarily of read speech, with a 5,000- to 20,000-word vocabulary. It is the current ARPA speech recognition research corpus. Compared to the RM corpus, it is a more challenging corpus for the neural net system due to the greater length of WSJ utterances and the higher perplexity of the WSJ task. So we would expect greater difficulty in improving performance on the WSJ corpus.

# 3    TRAINING CONTEXT-DEPENDENT MODELS WITH REGULARIZATION

## 3.1    WHY REGULARIZATION

In contrast to the context-independent modeling for the RM corpus, we are concentrating on context-dependent modeling for the WSJ corpus. In context-dependent modeling, instead of using a single neural net to recognize phonemes in all contexts, different neural networks are used to recognize phonemes in different contexts. Because of the paucity of training data for some context models, we found that we had an overfitting problem.

Regularization provides a class of smoothing techniques to ameliorate the overfitting problem [5]. We started using regularization in our initial one-layer sigmoidal neural network system. The regularization term added here is to regulate how far the context-dependent parameters can move away from their initial estimates, which are context-independent parameters. This is different from the usual weight decay technique in neural net literature, and it is designed specifically for our problem. The objective function is shown below:

$$-\frac{1}{N_d}\sum_c \underbrace{\left[\sum_{i \notin c} \log(1 - f_i) + \sum_{i \in c} \log f_i\right]}_{\text{Distortion measure Er(W)}} + \underbrace{\lambda \|\vec{W} - \vec{W}_0\|^2}_{\text{Regularization Term}} \tag{1}$$

where $f_i$ is the net output for class $i$, $\|W\|$ is the Euclidean norm of all weights in all the networks, $\|W_0\|$ is the initial estimate of weights from a context-independent neural network, $N_d$ is the number of data points. $\lambda$ is the regularization parameter which controls the tradeoff between the "smoothness" of the solution, as measured by $\|\vec{W} - \vec{W}_0\|^2$, and the deviation from the data as measured by the distortion.

The optimal $\lambda$, which gives the best generalization to a test set, can be estimated by generalized cross-validation [5]. If the distortion measure as shown in (2)

$$\frac{1}{N_d}\|AW - b\|^2 + \lambda \|W\|^2 \tag{2}$$

is a quadratic function in terms of network weights $W$, the optimal $\lambda$ is that which gives the minimum of a generalized cross-validation index $V(\lambda)$ [6]:

$$V(\lambda) = \frac{\frac{1}{N_d}\|A(\lambda) - b\|^2}{1 - \frac{1}{N_d}tr(A(\lambda))} \tag{3}$$

where $A(\lambda) = A(A^T A + N_d \lambda I)A^T$. $V(\lambda)$ is an easily calculated function based on singular value decomposition (SVD):

$$V(\lambda) = \frac{N_d \left[\|b\|^2 - \sum_{j=1}^p (\frac{d_j^2}{d_j^2 + N_d \lambda})^2 z_j^2\right]}{\left[N_d - \sum_{j=1}^p \frac{d_j^2}{d_j^2 + N_d \lambda}\right]^2} \tag{4}$$

where $A = UDV^T$, singular decomposition of A, $z = U^T b$. Figure 2 shows an example plot of $V(\lambda)$. A typical optimal $\lambda$ has an inverse relation to the number of samples in each class, indicating that $\lambda$ is gradually reduced with the presence of more data.

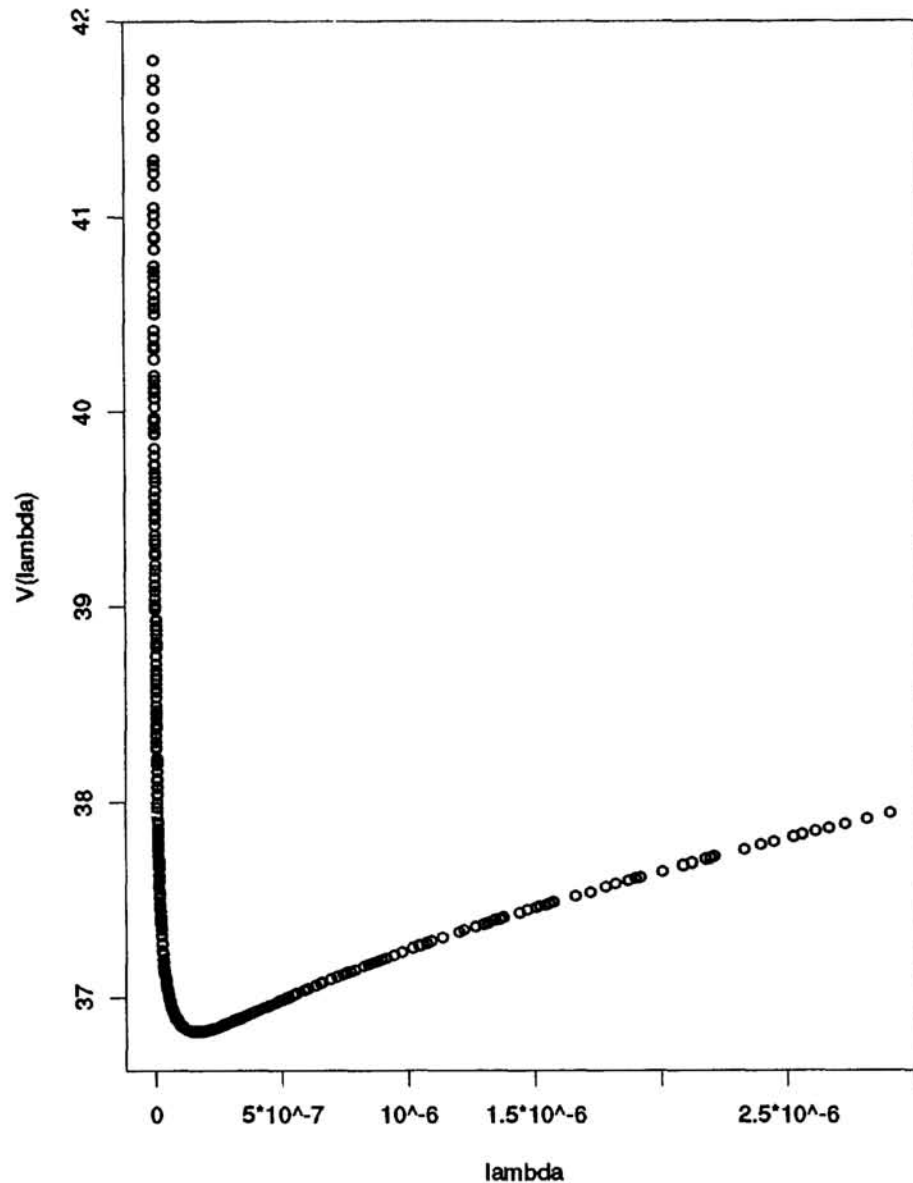

Figure 2: A typical $V(\lambda)$

Just as the linear least squares method can be generalized to a nonlinear least squares problem by an iterative procedure, so selecting the optimal value of the regularization parameter in a quadratic error criterion can be generalized to a non-quadratic error criterion iteratively. We developed an iterative procedure to apply the cross-validation technique to a non-quadratic error function, for example, the cross-entropy criterion $Er(W)$ in (1) as follows:

1. Compute distortion $Er(W_n)$ for an estimate $W_n$.

2. Compute gradient $g_n$ and Hessian $H_n$ of the distortion $Er(W_n)$.

3. Compute the singular value decomposition of $H_n = V_n^T D_n V_n$. Set $z_n = V_n g_n$.

4. Evaluate a generalized cross-validation index $V_n(\lambda)$ similar to (2) as follows, for a range of $\lambda$'s and select the $\lambda_n$ that gives the minimum $V_n$.

$$V_n(\lambda) = \frac{N_d \left[ Er(W_n) - \sum_j \frac{\frac{1}{2}d_j + N_d\lambda}{(d_j + N_d\lambda)^2} z_n^2 \right]}{\left[ N_d - \sum_j \frac{d_j}{d_j + N_d\lambda} \right]^2} \tag{5}$$

5. Set $W_{n+1} = W_n - (H_n + N_d\lambda_n)^{-1}g_n$.

6. Go to 1 and iterate.

Note that $\lambda$ is adjusted at each iteration. The final value of $\lambda_n$ is taken as the optimal $\lambda$. Iterative regularization parameter selection shows that $\lambda$ converges, for example, to $\frac{10^{-5}}{N_d}$ from one of our experiments.

## 3.2   A TWO-LAYER NEURAL NETWORK SYSTEM WITH REGULARIZATION

We then extended our regularization work from the one-layer sigmoidal network system to a two-layer sigmoidal network system. The first layer of the network works as a feature extractor and is shared by all phonetic classes. Theoretically, in order to benefit from its larger capability of representing phonetic segments, the number of hidden units of a two-layer network should be much greater than the number of input dimensions. However, a large number of hidden units can cause serious overfitting problems when the number of training samples is less than the number of parameters for some context models. Therefore, regularization is more useful here. Because the second layer can be trained as a one-layer net, the regularization techniques we developed for a one-layer net can be applied here to train the second layer.

In our implementation, a weighted least squares error measure was used at the output layer. First, the weights for the two-layer system were initialized with random numbers between -1 and 1. Fixing the weights for the second layer, we trained the first layer by using gradient descent; then fixing the weights for the first layer, we trained the second layer by linear least squares with a regularization term, without the sigmoidal function at the output. We stopped after one iteration for our initial experiment.

## 4   TRAINING SNN WITH PROJECTION PURSUIT

### 4.1   WHY PROJECTION PURSUIT

As we described in the previous section, regularization is especially useful in training the second layer of a two-layer network. In order to take advantage of the two-layer layer structure, we want to train the first layer as well. However, once the number of the hidden units is large, the number of weights in the first layer is huge, which makes the first layer very difficult to train. Projection pursuit presents a useful technique to use a large hidden layer but still keep the number of weights in the first layer as small as possible.

The original projection pursuit is a nonparametric statistical technique to find interesting low dimensional projections of high dimensional data sets [7]. The parametric version of it, a projection pursuit learning network (PPLN) has a structure very similar to a two-layer sigmoidal network network [7]. In a traditional two-layer neural network, the weights in the first layer can be viewed as hyperplanes in the input space. It has been proposed that a special function of the first layer is to partition the input space into cells through these hyperplanes [8]. The second layer groups these cells together to form decision regions.

The accuracy or resolution of the decision regions is completely specified by the size and density of the cells which is determined by the number and placement of the first layer hyperplanes in the input space.

In a two-layer neural net, since the weights in the first layer can go anywhere, there are no restrictions on the placement of these hyperplanes. In contrast, a projection pursuit learning network restricts these hyperplanes in some major "interesting" directions. In other words, hidden units are grouped into several distinct directions. Of course, with this grouping, the number of cells in the input space is reduced somewhat. However, the interesting point here is that this restriction does not reduce the number of cells asymptotically [7]. In other words, grouping hidden units does not affect the number of cells much. Consequently, for a fixed number of hidden units, the number of parameters in the first layer in a projection pursuit learning network is much less than in a traditional neural network. Therefore, a projection pursuit learning network is easier to train and generalizes better.

### 4.2  HOW TO TRAIN A PPLN

In our implementation, the distinct projection directions were shared by all context-dependent models, and they were trained context-independently. We then trained these direction parameters with back-propagation. The second layer was trained with regularization. Iterations can go back and forth between the two layers.

## 5  COMBINATIONS OF DIFFERENT MODELS

In the last two sections, we talked about using regularization and projection pursuit to optimize our neural network system. In this section, we will discuss another optimization method, combining different models into a hybrid system. The combining method is based on the N-best rescoring paradigm [2].

The N-best rescoring paradigm is a mechanism that allows us to build a hybrid system by combining different knowledge sources. For example, in the RM corpus, we successfully combined the HMM system, the SNN system and word-pair grammar into a single hybrid system which achieved the state-of-the-art. We have been using this N-best rescoring paradigm to combine different models in the WSJ corpus as well. These different models include SNN left context, right context, and diphone models, HMM models, and a language model known as statistical grammar. We will show how to obtain a reasonable combination of different systems from Bayes rule.

The goal is to compute $P(S|X)$, the probability of the sentence $S$ given the observation sequence $X$. From Bayes rule,

$$
\begin{aligned}
P(S|X)_{SNN} &= P(S)\frac{P(X|S)}{p(X)} \\
&\approx P(S)\prod_x \frac{P(x|S)}{P(x)} \\
&\approx P(S)\prod_x \frac{P(x|p,c)}{P(x)} \\
&\approx P(S)\prod_x \frac{P(p|x,c)}{P(p|c)}
\end{aligned}
$$

where $X$ is a sequence of acoustic features $x$ in each phonetic segment; $p$ and $c$ is the phoneme class and context for the segment, respectively. The following three approximations are used here:

- $P(X|S) = \prod_x P(x|S)$.
- $P(x|S) = P(x|p, c)$.
- $P(c|x) = P(c)$.

Therefore, in a SNN system, we use the following approximation from Bayes rule:

$$P(S|X)_{NN} \approx P(S) \prod_x \frac{P(p|x, c)}{P(p|c)}$$

where

$P(S)$: Word grammar score.

$\prod_x P(p|x, c)$: Neural net score.

$\prod_x P(p|c)$: Phone grammar score.

These three scores together with HMM scores are combined in the SNN/HMM hybrid system.

# 6   EXPERIMENTAL RESULTS

|  | Development Set | Nov92 Test |
|---|---|---|
| HMM | 11.0 | 8.5 |
| Baseline SNN | 11.7 | – |
| Regularization and Projection Pursuit SNN | 11.2 | 9.1 |
| Baseline SNN/HMM | 10.3 | 7.7 |
| Regularization and Projection Pursuit SNN/HMM | 9.5 | 7.2 |

Table 1: Word Error Rates for 5K, Bigram Grammar

|  | Development Set | Nov93 Test |
|---|---|---|
| HMM | 14.4 | 14.0 |
| Regularization and Projection Pursuit SNN | 14.6 | – |
| Regularization and Projection Pursuit SNN/HMM | 13.0 | 12.3 |

Table 2: Word Error Rates for 20K, Trigram Grammar

Speaker-independent CSR tests were performed on the 5,000-word (5K) and 20,000-word (20K) ARPA Wall Street Journal corpus. Bigram and trigram statistical grammars were used. The basic neural network structure consists of 80 inputs, 500 hidden units and 46 outputs. There are 125 projection directions in the first layer. Context models consist of

right context models and left diphone models. In the right context models, we used 46 different networks to recognize each phoneme in each of the different right contexts. In the left diphone models, a segment input consisted of the first half segment of the current phone plus the second half segment of the previous phone. Word error rates are shown in Tables 1 and 2.

Comparing the first two rows of Table 1 and Table 2, we can see that the two-layer neural network system alone is at the level of state-of-the-art HMM systems. Shown in Row 3 and 5 of Table 1, regularization and projection pursuit improve the performance of neural net system. The hybrid SNN/HMM system reduces the word error rate 10%-15% over the HMM system in both tables.

# 7   CONCLUSIONS

Neural net technology is useful in advancing the state-of-the-art in continuous speech recognition system. Optimization methods, like regularization and projection pursuit, improve the performance of the neural net system. Our hybrid SNN/HMM system reduces the word error rate 10%-15% over the HMM system on 5,000-word and 20,000-word WSJ corpus.

**Acknowledgments**

This work was funded by the Advanced Research Projects Agency of the Department of Defense.

# References

[1] M. Cohen, H. Franco, N. Morgan, D. Rumelhart and V. Abrash, "Context-Dependent Multiple Distribution Phonetic Modeling with MLPS", in em Advances in Neural Information Processing Systems 5, eds. S. J. Hanson, J. D. Cowan and C. L. Giles. Morgan Kaufmann Publishers, San Mateo, 1993.

[2] G. Zavaliagkos, Y. Zhao, R. Schwartz and J. Makhoul, " A Hybrid Neural Net System for State-of-the-Art Continuous Speech Recognition", in em Advances in Neural Information Processing Systems 5, eds. S. J. Hanson, J. D. Cowan and C. L. Giles. Morgan Kaufmann Publishers, San Mateo, 1993.

[3] A. Barron, "Statistical properties of artificial neural networks," *IEEE Conf. Decision and Control*, Tampa, FL, pp. 280-285, 1989.

[4] H. Gish, "A probabilistic approach to the understanding and training of neural network classifiers," *IEEE Int. Conf. Acoust., Speech, Signal Processing*, April 1990.

[5] G. Wahba, *Spline Models for Observational Data*, CBMS-NSF Regional Conference Series in Applied Mathematics, 1990.

[6] D. M. Bates, M. J. Lindstrom, G. Wahba and B. S. Yandell, "GCVPACK–Routines for Generalized Cross Validation", *Comm. Statist.- Simula.*, 16(4), 1247-1253 (1987).

[7] Y. Zhao and C. G. Atkeson, "Implementing Projection Pursuit Learning", to appear in *Neural Computation*, in preparation.

[8] J. Makhoul, A. El-Jaroudi and R. Schwartz, "Partitioning Capabilities of Two-layer Neural Networks", *IEEE Transactions on Signal Processing*, 39, pp. 1435-1440, 1991.
